# Eye Movements for Reward Maximization

**Nathan Sprague**
Computer Science Department
University of Rochester
Rochester, NY 14627
*sprague@cs.rochester.edu*

**Dana Ballard**
Computer Science Department
University of Rochester
Rochester, NY 14627
*dana@cs.rochester.edu*

## Abstract

Recent eye tracking studies in natural tasks suggest that there is a tight link between eye movements and goal directed motor actions. However, most existing models of human eye movements provide a bottom up account that relates visual attention to attributes of the visual scene. The purpose of this paper is to introduce a new model of human eye movements that directly ties eye movements to the ongoing demands of behavior. The basic idea is that eye movements serve to reduce uncertainty about environmental variables that are task relevant. A value is assigned to an eye movement by estimating the expected cost of the uncertainty that will result if the movement is *not* made. If there are several candidate eye movements, the one with the highest expected value is chosen. The model is illustrated using a humanoid graphic figure that navigates on a sidewalk in a virtual urban environment. Simulations show our protocol is superior to a simple round robin scheduling mechanism.

## 1 Introduction

This paper introduces a new framework for understanding the scheduling of human eye movements. The human eye is characterized by a small, high resolution fovea. The importance of foveal vision means that fast ballistic eye movements called saccades are made at a rate of approximately three per second to direct gaze to relevant areas of the visual field. Since the location of the fovea provides a powerful clue to what information the visual system is processing, understanding the scheduling and targeting of eye movements is key to understanding the organization of human vision.

The recent advent of portable eye-trackers has made it possible to study eye movements in everyday behaviors. These studies show that behaviors such as driving [1, 2] or navigating a city sidewalk [3] show rapid alternating saccades to different targets indicative of competing perceptual demands.

This paper introduces a model of how humans select visual targets in terms of the value of the information obtained. Previous work has modeled the direction of the eyes to targets primarily in terms of visual saliency [4]. Such models fail to incorporate the role of task demands and do not address the problem of resource contention. In contrast, our underlying premise is that much of routine human behavior can be understood in the framework of reward maximization. In other words, humans choose actions by trading off the cost of the

actions versus their benefits. Experiments show that the extent to which humans can make such trade-offs is very refined [5]. To keep track of the value of future real rewards such as money or calories, humans use internal chemical rewards such as dopamine [6].

One obvious way of modeling eye movement selection is to use a reinforcement learning strategy directly. However, standard reinforcement learning algorithms are are best suited to handling actions that have direct consequences for a task. Actions such as eye movements are more difficult to put in a reinforcement learning framework because they have indirect consequences: they do not change the state of the environment; they serve only to obtain information. We show a way of overcoming this difficulty while preserving the notion of reward maximization in the scheduling of eye movements. The basic idea is that eye movements serve to reduce uncertainty about environmental variables that are relevant to behavior. A value is assigned to an eye movement by estimating the expected cost of the uncertainty that will result if the movement is *not* made. If there are several candidate eye movements, the one with the highest potential loss is chosen.

We demonstrate these ideas through the example of a virtual human navigating through a rendered environment. The agent is faced with multiple simultaneous goals including walking along a sidewalk, picking up litter, and avoiding obstacles. He must schedule simulated eye movements so as to maximize his reward across the set of goals. We model eye movements as abstract sensory actions that serve to retrieve task relevant information from the environment. Our focus is on temporal scheduling; we are not concerned with the spatial targeting of eye movements. The purpose of this paper is to recast the question of how eye movements are scheduled, and to propose a possible answer. Experiments on real humans will be required to determine if this model accurately describes human behavior.

## 2   Learning Visually Guided Behaviors

Our model of visual control is built around the concept of visual behaviors. Here we borrow the usage of behavior from the robotics community to refer to a sensory-action control module that is responsible for handling a single narrowly defined goal [7]. The key advantage of the behavior based approach is compositionality: complex control problems can be solved by sequencing and combining simple behaviors. For the purpose of modeling human performance it is assumed that each behavior has the ability to direct the eye, perform appropriate visual processing to retrieve the information necessary for performance of the behavior's task, and choose an appropriate course of action.

As long as only one goal is active at a time the behavior based approach is straightforward: the appropriate behavior is put in control and has all the machinery necessary to pursue the goal. However it is often the case that multiple goals must be addressed at once. In this case there is need for arbitration mechanisms to distribute control among the set of active behaviors. In the following sections we will describe how physical control is arbitrated, and building on that framework, how eye movements are arbitrated.

Our approach to designing behaviors is to model each behavior's task as a Markov decision process and then find good policies using reinforcement learning. An MDP is described by a 4-tuple $(\mathcal{S}, \mathcal{A}, T, R)$, where $\mathcal{S}$ is the state space, $\mathcal{A}$ is the action space, and $T(s, a, s')$ is the transition function that indicates the probability of arriving in state $s'$ when action $a$ is taken in state $s$. The reward function $R(s, a)$ denotes the expected one-step payoff for taking action $a$ in state $s$. The goal of reinforcement learning algorithms is to discover an optimal policy $\pi^*(s)$ that maps states to actions so as to maximize discounted long term reward. Generally, we do not assume prior knowledge of $R$ and $T$.

One approach to finding optimal policies for MDPs is based on discovering the optimal value function $Q(s, a)$. This function denotes the expected discounted return if action $a$ is

taken in state $s$ and the optimal policy is followed thereafter. If $Q(s,a)$ is known then the learning agent can behave optimally by always choosing $\arg\max_a Q(s,a)$.

There are a number of algorithms for learning $Q(s,a)$ [8, 9] the simplest is to take random actions in the environment and use the Q-learning update rule:

$$Q(s,a) \leftarrow (1-\alpha)Q(s,a) + \alpha(r + \gamma \max_{a'} Q(s',a'))$$

Here $\alpha$ is a learning rate parameter, and $\gamma$ is a term that determines how much to discount future reward. As long as each state-action pair is visited infinitely often in the limit, this update rule is guaranteed to converge to the optimal value function.

A benefit of knowing the value function for each behavior is that the Q-values can be used to handle the arbitration problem. Here we assume that the behaviors share an action space. In order to choose a compromise action, it is assumed that the $Q$-function for the composite task is approximately equal to the sum of the $Q$-functions for the component tasks:

$$Q(s,a) \approx \sum_{i=1}^{n} Q_i(s_i,a), \tag{1}$$

where $Q_i(s_i,a)$ represents the $Q$-function for the $i$th active behavior. The idea of using Q-values for multiple goal arbitration was independently introduced in [10] and [11].

The real world interactions that this model is meant to address are best expressed through continuous rather than discrete state variables. The theoretical foundations of value based continuous state reinforcement learning are not as well established as for the discrete state case. However empirical results suggest that good results can be obtained by using a function approximator such as a CMAC along with the Sarsa(0) learning rule: [12]

$$Q(s,a) \leftarrow (1-\alpha)Q(s,a) + \alpha(r + \gamma Q(s',a'))$$

This rule is nearly identical to the Q-learning rule, except that the max action is replaced by the action that is actually observed on the next step. The Q-functions used throughout this paper are learned using this approach. For reasons of space this paper will not include a complete description of the training procedure used to obtain the Q-functions for the sidewalk task. More details can be found in [13] and [14].

## 3 A Composite Task: Sidewalk Navigation

The components of the sidewalk navigation task are to stay on the sidewalk, avoid obstacles, and pick up litter. This was chosen as a good example of a task with multiple goals and conflicting demands.

Our sidewalk navigation model has three behaviors, sidewalk following, obstacle avoidance, and litter collection. These behaviors share an action space composed of three actions: $15^o$ right turn, $15^o$ left turn, and no turn (medium gray, dark gray, and light gray arrows in Figure 1). During the sidewalk navigation task the virtual human walks forward at a steady rate of 1.3 meters per second. Every 300ms a new action is selected according to the action selection mechanism summarized in Equation (1).

Each of the three behaviors has a two dimensional state space. For obstacle avoidance the state space is comprised of the distance and angle, relative to the agent, to the nearest obstacle. The litter collection behavior uses the same parameterization for the nearest litter item. For the sidewalk following behavior the state space is the angle of the center-line of the sidewalk relative to the agent, as well as the signed distance to the center of the sidewalk, where positive values indicate that the agent is to the left of the center, and negative values indicate that the agent is to the right. All behaviors use the log of distance in order to

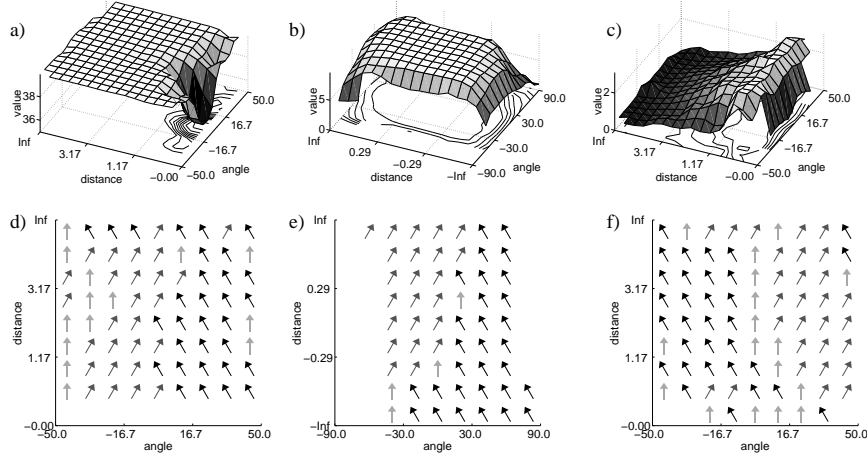

Figure 1: Q-values and policies for the three behaviors. Figures a)-c) show $\max_a Q(s, a)$ for the three behaviors: a) obstacle avoidance, b) sidewalk following and c) litter collection. Figures d)-f) show the corresponding policies for the three behaviors. Empty regions indicate areas that were not seen often enough during training to compute reliable values.

devote more of the state representation to areas near the agent. The agent receives two units of reward for every item of litter collected , one unit for every time step he remains on the sidewalk, and four units for every time step he does not collide with an obstacle. Figure 1 shows a representation of the Q-functions and policies for the three behaviors.

The behaviors use simple sensory routines to retrieve the relevant state information from the environment. The sidewalk following behavior searches for pixels at the border of the sidewalk and the grass, and finds the most prominent line using a hough transform. The litter collection routine uses color based matching to find the location of litter items. The obstacle avoidance routines refers to the world model directly to compute a rough depth map of the area ahead, and from that extracts the position of the nearest obstacle.

## 4   Eye Movements and Internal Models

The discussion above assumed that the MDPs have perfect state information. In order to model limited sensory capacity this assumption must be weakened. Without perfect information the component tasks are most accurately described as partially observable MDPs.

The Kalman filter [15] solves the problem of tracking a discrete time, continuous state variable in the face of noise in both measurements and in the underlying process being tracked. It allows us to represent the consequences of not having the most recent information from an eye movement. The Kalman filter has two properties that are important in this respect. One is that it not only maintains an estimate of the state variable, it also maintains an estimate of the uncertainty. With this information the behaviors may treat their state estimates as continuous random variables with known probability distributions. The other useful property of the Kalman filter is that it is able to propagate state estimates in the absence of sensory information. The state estimate is updated according to the system dynamics, and the uncertainty in the estimate increases according to the known process noise.

In order to simulate the fact that only one area of the visual field may be foveated, only one behavior is allowed access to perception during each 300ms time step. That behavior updates its Kalman filter with a measurement, while the others propagate their estimates

and track the increase in uncertainty. In order to simulate noise in the estimator, the state estimates are corrupted with zero-mean normally distributed noise at each time step.

Since the agent does not have perfectly up to date state information, he must select the best action given his current estimates of the state. A reasonable way of selecting an action under uncertainty is to select the action with the highest expected return. Building on Equation (1) we have the following: $a_E = \arg\max_a E[\sum_{i=1}^n Q_i(s_i, a)]$, where the expectation is computed over the state variables for the behaviors. By distributing the expectation, and making a slight change to the notation we can write this as:

$$a_E = \arg\max_a \sum_{i=1}^n Q_i^E(s_i, a), \tag{2}$$

where $Q_i^E$ refers to the expected $Q$-value of the $i$th behavior. In practice we will estimate expectations by sampling from the distributions provided by the Kalman filter.

Selecting the action with the highest expected return does not guarantee that the agent will choose the best action for the true state of the environment. Whenever the agent chooses an action that is sub-optimal for the true state of the environment, he can expect to lose some return. We can estimate the expected loss as follows:

$$loss = E[\max_a \sum Q_i(s_i, a)] - E[\sum Q_i(s_i, a_E)]. \tag{3}$$

The term on the left-hand side of the minus sign expresses the expected return that the agent would receive if he were able to act with knowledge of the true state of the environment. The term on the right expresses the expected return if the agent is forced to choose an action based on his state estimate. The difference between the two can be thought of as the cost of the agent's current uncertainty. This value is guaranteed to be positive, and may be zero if all possible states would result in the same action choice.

The total expected loss does not help to select which of the behaviors should be given access to perception. To make this selection, the loss value needs to be broken down into the losses associated with the uncertainty for each particular behavior $b$:

$$loss_b = E\left[ \max_a \left( Q_b(s_b, a) + \sum_{i \in B, i \neq b} Q_i^E(s_i, a) \right) \right] - \sum_i Q_i^E(s_i, a_E). \tag{4}$$

Here the expectation on the left is computed only over $s_b$. The value on the left is the expected return if $s_b$ were known, but the other state variables were not. The value on the right is the expected return if none of the state variables are known. The difference is interpreted as the cost of the uncertainty associated with $s_b$.

Given that the $Q$ functions are known, and that the Kalman filters provide distributions over the state variables, it is straightforward to estimate $loss_b$ for each behavior $b$ by sampling. This value is then used to select which behavior will make an eye movement.

Figure 2 gives an example of several steps of the sidewalk task, the associated eye movements, and the state estimates. The eye movements are allocated to reduce the uncertainty where it has the greatest potential negative consequences for reward. For example, the agent fixates the obstacle as he draws close to it, and shifts perception to the other two behaviors when the obstacle has been safely passed.

It is important to recognize that the procedures outlined above for selecting actions and allocating perception are only approximations. Since the Q-tables were trained under the assumption of perfect state information, they will be somewhat inaccurate under conditions of partial observability. Note also that the behaviors actually employ multiple Kalman filters. For example if the obstacle avoidance behavior sees two obstacles it will initialize a filter for each. However, only the single closest object is used to determine the state for the purpose of action selection and scheduling eye movements.

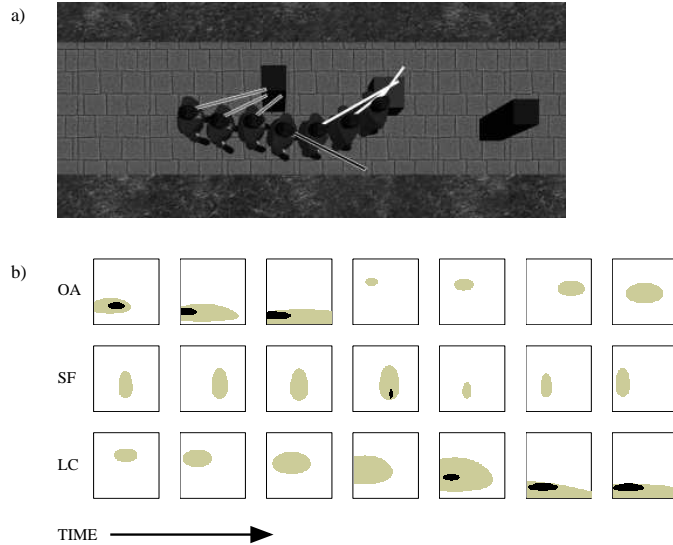

Figure 2: a) An overhead view of the virtual agent during seven time steps of the sidewalk navigation task. The two darker cubes are obstacles, and the lighter cube is litter. The rays projecting from the agent represent eye movements; gray rays correspond to obstacle avoidance, black rays correspond to sidewalk following, and white correspond to litter collection. b) State estimates during the same seven time steps. The top row shows the agent's estimates of the obstacle location. The axes here are the same as those presented in Figure 1. The light gray regions correspond to the 90% confidence bounds before any perception has taken place. When present, the black regions correspond to the 90% confidence bounds after an eye movement has been made. The second and third rows show the corresponding information for the sidewalk following and litter collection tasks.

## 5  Results

In order to test the effectiveness of the loss minimization approach, we compare it to two alternative scheduling mechanisms: round robin, which sequentially rotates through the three behaviors, and random, which makes a uniform random selection on each time step. Round robin might be expected to perform well in this task, because it is optimal in terms of minimizing long waits across the three behaviors.

The three strategies are compared under three different conditions. In the default condition exactly one behavior is given access to perception on each time step. The other two conditions investigate the performance of the system under increasing perceptual load. During these trials 33% or 66% of steps are randomly selected to have no perceptual action at all.

For the default condition the average per-step reward is .034 higher for the loss minimization scheduling than for the round robin scheduling. Two factors make this difference more substantial than it first appears. The first is that the reward scale for this task does not start at zero: when taking completely random actions the agent receives an average of 4.06 units of reward per step. Therefore the advantage of the loss minimization approach is a full 3.6% over round robin, relative to baseline performance.

The second factor to consider is the sheer number of eye movements that a human makes over the course of a day: a conservative estimate is 150,000. The average benefit of properly scheduling a single eye movement may be small, but the cumulative benefit is enormous. To

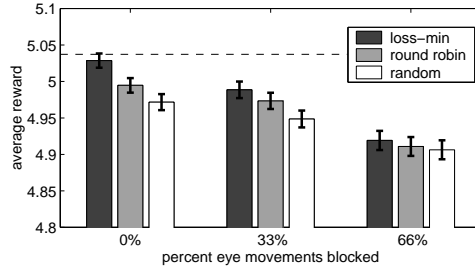

Figure 3: Comparison of loss minimization scheduling to round robin and random strate-
gies. For each condition the agent is tested for 500 trials lasting 20 seconds each. In the
33% and 66% conditions the corresponding percentage of eye movements are randomly
blocked, and no sensory input is allowed. The error bars represent 95% confidence in-
tervals. The dashed line at 5.037 indicates the average reward received when all three
behaviors are given access to perception at each time step. This can be seen as an upper
bound on the possible reward.

make this point more concrete, notice that over a period of one hour of sidewalk navigation
the agent will lose around 370 units of reward if he uses round robin instead of the loss
minimization approach. In the currency of reward this is equal to 92 additional collisions
with obstacles, 184 missed litter items, or two additional minutes spent off the sidewalk.

Under increasing perceptual load the loss minimization strategy begins to lose its advantage
over the other two techniques. This could be because the Q-tables become increasingly
inaccurate as the assumption of perfect state information becomes less valid.

## 6   Related Work

The action selection mechanism from Equation (2) is essentially a continuous state version
of the Q-MDP algorithm for finding approximate solutions to POMDPs [16]. Many discrete
POMDP solution and approximation techniques are built on the idea of maintaining a belief
state, which is a probability distribution over the unobserved state variables. The idea
behind the Q-MDP algorithm is to first solve the underlying MDP, and then choose actions
according to $\arg\max_a \sum_s bel(s)Q(s,a)$, where $bel(s)$ is the probability that the system
is in state $s$ and $Q(s,a)$ is the optimal value function for the underlying MDP. The main
drawback of the Q-MDP algorithm is that it does not specifically seek out actions that
reduce uncertainty. In this work the Kalman filters serve precisely the role of maintaining
a continuous belief state, and the problem of reducing uncertainty is handled through the
separate mechanism of choosing eye movements to minimize loss.

The gaze control system introduced in [17] also addresses the problem of perceptual arbi-
tration in the face of multiple goals. The approach taken in that paper has many parallels to
the work presented here, although the focus is on robot control rather than human vision.

## 7   Discussion and Conclusions

Any system for controlling competing visuo-motor behaviors that all require access to a
sensor such as the human eye faces a resource allocation problem. Gaze cannot be two
places at once and therefore has to be shared among the concurrent tasks. Our model
resolves this difficulty by computing the cost of having inaccurate state information for
each active behavior. Reward can be maximized by allocating gaze to the behavior that

stands to lose the most. As the simulations show, the performance of the algorithm is superior both to the round robin protocol and to a random allocation strategy.

It is possible for humans to examine locations in the visual scene without overt eye movements. In such cases our formalism would still be relevant to the covert allocation of visual resources.

Finally, although the expected loss protocol is developed for eye movements, the computational strategy is very general and extends to any situation where there are multiple active behaviors that must compete for information gathering sensors.

## Acknowledgments

This material is based upon work supported by grant number P200A000306 from the Department of Education, grant number 5P41RR09283 from the National Institutes of Health and a grant number E1A-0080124 from the National Science Foundation.

## References

[1] M. F. Land and D. Lee. Where we look when we steer. *Nature*, 377, 1994.

[2] H. Shinoda, M. Hayhoe, and A. S Shrivastava. The coordination of eye, head, and hand movements in a natural task. *Vision Research*, 41, 2001.

[3] D. Ballard and N. Sprague. Attentional resource allocation in extended natural tasks [abstract]. *Journal of Vision*, 2(7):568a, 2002.

[4] L. Itti and C. Koch. Computational modeling of visual attention. *Nature Reviews Neuroscience*, 2(3):194–203, Mar 2001.

[5] L. Maloney and M. Landy. When uncertainty matters: the selection of rapid goal-directed movements [abstract]. *Journal of Vision*, (to appear).

[6] P. Waelti, A. Dickinson, and W. Schultz. Dopamine responses comply with basic assumptions of formal learning theory. *Nature*, 412, July 2001.

[7] Rodney A. Brooks. A robust layered control system for a mobile robot. *IEEE Journal of Robotics and Automation*, RA-2(1):14–23, April 1986.

[8] Leslie P. Kaelbling, Michael L. Littman, and Andrew W. Moore. Reinforcement learning: A survey. *Journal of Artificial Intelligence Research*, 4:237–285, 1996.

[9] R.S. Sutton and A.G. Barto. *Reinforcement Learning: An Introduction*. MIT Press, 1998.

[10] M. Humphrys. Action selection methods using reinforcement learning. In *Proceedings of the Fourth International Conference on Simulation of Adaptive Behavior*, 1996.

[11] J. Karlsson. *Learning to Solve Multiple Goals*. PhD thesis, University of Rochester, 1997.

[12] R. Sutton. Generalization in reinforcement learning: Successful examples using sparse coarse coding. In *Advances in Neural Information Processing Systems*, volume 8, 1996.

[13] N. Sprague and D. Ballard. Multiple-goal reinforcement learning with modular sarsa(0). In *International Joint Conference on Artificial Intelligence*, August 2003.

[14] N. Sprague and D. Ballard. Multiple goal learning for a virtual human. Technical Report 829, University Of Rochester Computer Science Department, 2004.

[15] R. E. Kalman. A new approach to linear filtering and prediction problems. *Transactions of the ASME–Journal of Basic Engineering*, 82(Series D):35–45, 1960.

[16] A. Cassandra. *Exact and approximate algorithms for partially observable Markov decision processes*. PhD thesis, Brown University, 1998.

[17] J. F. Seara, K. H. Strobl, E. Martin, and G. Schmidt. Task-oriented and situation-dependent gaze control for vision guided autonomous walking. In *Proceedings of the 3rd IEEE-RAS International Conference on Humanoid Robots*, 2003.
